# A Bayesian Framework for Cross-Situational Word-Learning

**Michael C. Frank, Noah D. Goodman, and Joshua B. Tenenbaum**
Department of Brain and Cognitive Science
Massachusetts Institute of Technology
`{mcfrank, ndg, jbt}@mit.edu`

## Abstract

For infants, early word learning is a chicken-and-egg problem. One way to learn a word is to observe that it co-occurs with a particular referent across different situations. Another way is to use the social context of an utterance to infer the intended referent of a word. Here we present a Bayesian model of cross-situational word learning, and an extension of this model that also learns which social cues are relevant to determining reference. We test our model on a small corpus of mother-infant interaction and find it performs better than competing models. Finally, we show that our model accounts for experimental phenomena including mutual exclusivity, fast-mapping, and generalization from social cues.

To understand the difficulty of an infant word-learner, imagine walking down the street with a friend who suddenly says "dax blicket philbin na fivy!" while at the same time wagging her elbow. If you knew any of these words you might infer from the syntax of her sentence that blicket is a novel noun, and hence the name of a novel object. At the same time, if you knew that this friend indicated her attention by wagging her elbow at objects, you might infer that she intends to refer to an object in a nearby show window. On the other hand if you already knew that "blicket" meant the object in the window, you might be able to infer these elements of syntax and social cues. Thus, the problem of early word-learning is a classic chicken-and-egg puzzle: in order to learn word meanings, learners must use their knowledge of the rest of language (including rules of syntax, parts of speech, and other word meanings) as well as their knowledge of social situations. But in order to learn about the facts of their language they must first learn some words, and in order to determine which cues matter for establishing reference (for instance, pointing and looking at an object but normally not waggling your elbow) they must first have a way to know the intended referent in some situations.

For theories of language acquisition, there are two common ways out of this dilemma. The first involves positing a wide range of innate structures which determine the syntax and categories of a language and which social cues are informative. (Though even when all of these elements are innately determined using them to learn a language from evidence may not be trivial [1].) The other alternative involves bootstrapping: learning some words, then using those words to learn how to learn more. This paper gives a proposal for the second alternative. We first present a Bayesian model of how learners could use a statistical strategy—cross-situational word-learning—to learn how words map to objects, independent of syntactic and social cues. We then extend this model to a true bootstrapping situation: using social cues to learn words while using words to learn social cues. Finally, we examine several important phenomena in word learning: mutual exclusivity (the tendency to assign novel words to novel referents), fast-mapping (the ability to assign a novel word in a linguistic context to a novel referent after only a single use), and social generalization (the ability to use social context to learn the referent of a novel word). Without adding additional specialized machinery, we show how these can be explained within our model as the result of domain-general probabilistic inference mechanisms operating over the linguistic domain.

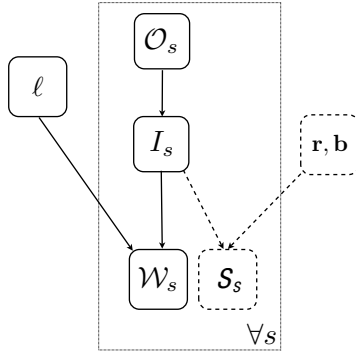

Figure 1: Graphical model describing the generation of words ($\mathcal{W}_s$) from an intention ($I_s$) and lexicon ($\ell$), and intention from the objects present in a situation ($\mathcal{O}_s$). The plate indicates multiple copies of the model for different situation/utterance pairs ($s$). Dotted portions indicate additions to include the generation of social cues $\mathcal{S}_s$ from intentions.

# 1   The Model

Behind each linguistic utterance is a meaning that the speaker intends to communicate. Our model operates by attempting to infer this intended meaning (which we call the *intent*) on the basis of the utterance itself and observations of the physical and social context. For the purpose of modeling early word learning—which consists primarily of learning words for simple object categories—in our model, we assume that intents are simply groups of objects.

To state the model formally, we assume the non-linguistic situation consists of a set $\mathcal{O}_s$ of objects and that utterances are unordered sets of words $\mathcal{W}_s$[1]. The lexicon $\ell$ is a (many-to-many) map from words to objects, which captures the *meaning* of those words. (Syntax enters our model only obliquely by different treatment of words depending on whether they are in the lexicon or not—that is, whether they are common nouns or other types of words.) In this setting the speaker's intention will be captured by a set of objects in the situation to which she intends to refer: $I_s \subseteq \mathcal{O}_s$. This setup is indicated in the graphical model of Fig. 1. Different situation-utterance pairs $\mathcal{W}_s, \mathcal{O}_s$ are independent given the lexicon $\ell$, giving:

$$P(\mathcal{W}|\ell, \mathcal{O}) = \prod_s \sum_{I_s} P(\mathcal{W}_s|I_s, \ell) \cdot P(I_s|\mathcal{O}_s). \tag{1}$$

We further simplify by assuming that $P(I_s|\mathcal{O}_s) \propto 1$ (which could be refined by adding a more detailed model of the communicative intentions a person is likely to form in different situations). We will assume that words in the utterance are generated independently given the intention and the lexicon and that the length of the utterance is observed. Each word is then generated from the intention set and lexicon by first choosing whether the word is a referential word or a non-referential word (from a binomial distribution of weight $\gamma$), then, for referential words, choosing which object in the intent it refers to (uniformly). This process gives:

$$P(\mathcal{W}_s|I_s, \ell) = \prod_{w \in \mathcal{W}_s} \left[ (1 - \gamma) P_{\text{NR}}(w|\ell) + \gamma \sum_{x \in I_s} \frac{1}{|I_s|} P_{\text{R}}(w|x, \ell) \right]. \tag{2}$$

The probability of word $w$ referring to object $x$ is $P_{\text{R}}(w|x, \ell) \propto \delta_{x \in \ell(w)}$, and the probability of word $w$ occurring as a non-referring word is

$$P_{\text{NR}}(w|\ell) \propto \begin{cases} 1 & \text{if } \ell(w) = \emptyset, \\ \kappa & \text{otherwise.} \end{cases} \tag{3}$$

(this probability is a distribution over all words in the vocabulary, not just those in lexicon $\ell$). The constant $\kappa$ is a penalty for using a word in the lexicon as a non-referring word—this penalty indirectly enforces a light-weight difference between two different groups of words (parts-of-speech): words that refer and words that do not refer.

Because the generative structure of this model exposes the role of speaker's intentions, it is straightforward to add non-linguistic social cues. We assume that social cues such as pointing are generated

from the speaker's intent independently of the linguistic aspects (as shown in the dotted arrows of Fig. 1). With the addition of social cues $\mathcal{S}_s$, Eq. 1 becomes:

$$P(\mathcal{W}|\ell,\mathcal{O}) = \prod_s \sum_{I_s} P(\mathcal{W}_s|I_s,\ell) \cdot P(\mathcal{S}_s|I_s) \cdot P(I_s|\mathcal{O}_s). \tag{4}$$

We assume that the social cues are a set $S_i(x)$ of independent binary (cue present or not) feature values for each object $x \in \mathcal{O}_s$, which are generated through a noisy-or process:

$$P(S_i(x){=}1|I_s,r_i,b_i) = 1 - (1-b_i)(1-r_i)^{\delta_{x \in I_s}}. \tag{5}$$

Here $r_i$ is the *relevance* of cue $i$, while $b_i$ is its *base rate*.

For the model without social cues the posterior probability of a lexicon given a set of situated utterances is:

$$P(\ell|\mathcal{W},\mathcal{O}) \propto P(\mathcal{W}|\ell,\mathcal{O})P(\ell). \tag{6}$$

And for the model with social cues the joint posterior over lexicon and cue parameters is:

$$P(\ell,\mathbf{r},\mathbf{b}|\mathcal{W},\mathcal{O}) \propto P(\mathcal{W}|\ell,\mathbf{r},\mathbf{b},\mathcal{O})P(\ell)P(\mathbf{r},\mathbf{b}). \tag{7}$$

We take the prior probability of a lexicon to be exponential in its size: $P(\ell) \propto e^{-\alpha|\ell|}$, and the prior probability of social cue parameters to be uniform.

Given the model above and the corpus described below, we found the best lexicon (or lexicon and cue parameters) according to Eq. 6 and 7 by MAP inference using stochastic search[2].

## 2 Previous work

While cross-situational word-learning has been widely discussed in the empirical literature, e.g., [2], there have been relatively few attempts to model this process computationally. Siskind [3] created an ambitious model which used deductive rules to make hypotheses about propositional word meanings their use across situations. This model achieved surprising success in learning word meanings in artificial corpora, but was extremely complex and relied on the availability of fully coded representations of the meaning of each sentence, making it difficult to extend to empirical corpus data. More recently, Yu and Ballard [4] have used a machine translation model (similar to IBM Translation Model I) to learn word-object association probabilities. In their study, they used a pre-existing corpus of mother-infant interactions and coded the objects present during each utterance (an example from this corpus—illustrated with our own coding scheme—is shown in Fig. 2). They applied their translation model to estimate the probability of an object given a word, creating a table of associations between words and objects. Using this table, they extracted a lexicon (a group of word-object mappings) which was relatively accurate in its guesses about the names of objects that were being talked about. They further extended their model to incorporate prosodic emphasis on words (a useful cue which we will not discuss here) and joint attention on objects. Joint attention was coded by hand, isolating a subset of objects which were attended to by both mother and infant. Their results reflected a sizable increase in recall with the use of social cues.

## 3 Materials and Assessment Methods

To test the performance of our model on natural data, we used the Rollins section of the CHILDES corpus[5]. For comparison with the model by Yu and Ballard [4], we chose the files me03 and di06, each of which consisted of approximately ten minutes of interaction between a mother and a pre-verbal infant playing with objects found in a box of toys. Because we were not able to obtain the exact corpus Yu and Ballard used, we recoded the objects in the videos and added a coding of social cues co-occurring with each utterance. We annotated each utterance with the set of objects visible to the infant and with a social coding scheme (for an illustrated example, see Figure 2). Our social code included seven features: infants eyes, infants hands, infants mouth, infant touching, mothers hands, mothers eyes, mother touching. For each utterance, this coding created an object by social feature matrix.

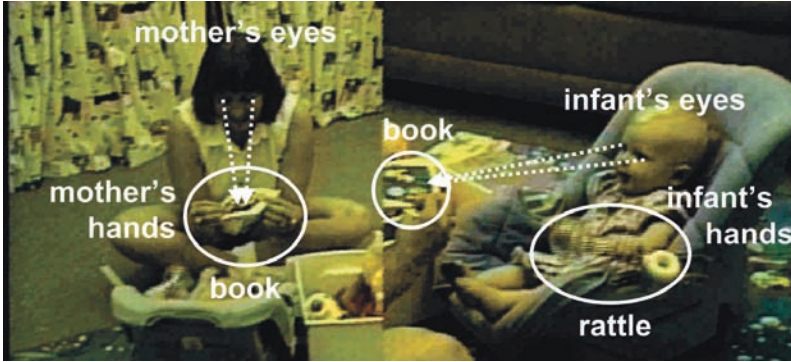

Figure 2: A still frame from our corpus showing the coding of objects and social cues. We coded all mid-sized objects visible to the infant as well as social information including what both mother and infant were touching and looking at.

We evaluated all models based on their coverage of a gold-standard lexicon, computing precision (how many of the word-object mappings in a lexicon were correct relative to the gold-standard), recall (how many of the total correct mappings were found), and their geometric mean, F-score. However, the gold-standard lexicon for word-learning is not obvious. For instance, should it include the mapping between the plural "pigs" or the sound "oink" and the object PIG? Should a gold-standard lexicon include word-object pairings that are correct but were not present in the learning situation? In the results we report, we included those pairings which would be useful for a child to learn (e.g., "oink" → PIG) but not including those pairings which were not observed to co-occur in the corpus (however, modifying these decisions did not affect the qualitative pattern of results).

## 4    Results

For the purpose of comparison, we give scores for several other models on the same corpus. We implemented a range of simple associative models based on co-occurrence frequency, conditional probability (both word given object and object given word), and point-wise mutual information. In each of these models, we computed the relevant statistic across the entire corpus and then created a lexicon by including all word-object pairings for which the association statistic met a threshold value. We additionally implemented a translation model (based on Yu and Ballard [4]). Because Yu and Ballard did not include details on how they evaluated their model, we scored it in the same way as the other associative models, by creating an association matrix based on the scores $P(O|W)$ (as given in equation (3) in their paper) and then creating a lexicon based on a threshold value. In order to simulate this type of threshold value for our model, we searched for the MAP lexicon over a range of parameters $\alpha$ in our prior (the larger the prior value, the less probable a larger lexicon, thus this manipulation served to create more or less selective lexicons) .

**Base model.** In Figure 3, we plot the precision and the recall for lexicons across a range of prior parameter values for our model and the full range of threshold values for the translation model and two of the simple association models (since results for the conditional probability models were very similar but slightly inferior to the performance of mutual information, we did not include them). For our model, we averaged performance at each threshold value across three runs of 5000 search iterations each. Our model performed better than any of the other models on a number of dimensions (best lexicon shown in Table 1), both achieving the highest F-score and showing a better tradeoff between precision and recall at sub-optimal threshold values. The translation model also performed well, increasing precision as the threshold of association was raised. Surprisingly, standard co-occurrence statistics proved to be relatively ineffective at extracting high-scoring lexicons: at any given threshold value, these models included a very large number of incorrect pairs.

Table 1: The best lexicon found by the Bayesian model ($\alpha$=11, $\gamma$=0.2, $\kappa$=0.01).

| | | | | |
|---|---|---|---|---|
| baby → book | bigbird → bird | bird → rattle | birdie → duck | book → book |
| hand → hand | hat → hat | meow → kitty | moocow → cow | oink → pig |
| | on → ring | ring → ring | sheep → sheep | |

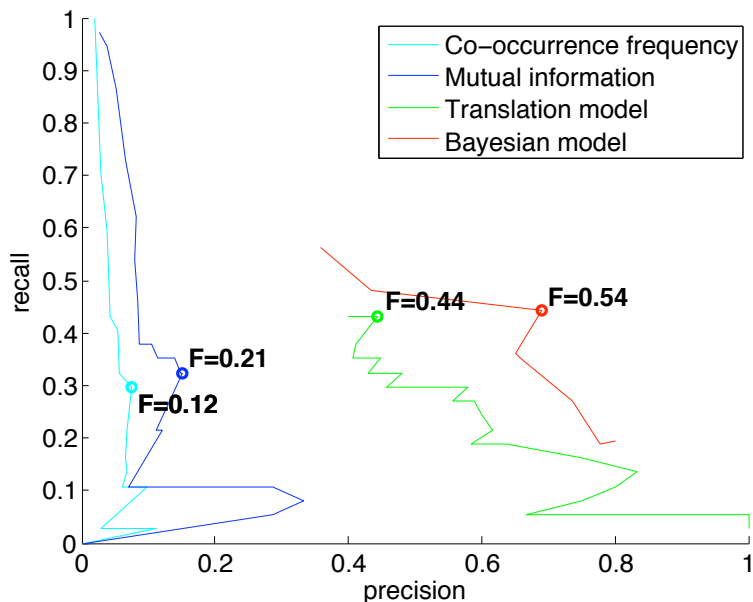

Figure 3: Comparison of models on corpus data: we plot model precision vs. recall across a range of threshold values for each model (see text). Unlike standard ROC curves for classification tasks, the precision and recall of a lexicon depends on the entire lexicon, and irregularities in the curves reflect the small size of the lexicons).

One additional virtue of our model over other associative models is its ability to determine which objects the speaker intended to refer to. In Table 2, we give some examples of situations in which the model correctly inferred the objects that the speaker was talking about.

**Social model.** While the addition of social cues did not increase corpus performance above that found in the base model, the lexicons which were found by the social model did have several properties that were not present in the base model. First, the model effectively and quickly converged on the social cues that we found subjectively important in viewing the corpus videos. The two cues which were consistently found relevant across the model were (1) the target of the infant's gaze and (2) the caregiver's hand. These data are especially interesting in light of the speculation that infants initially believe their own point of gaze is a good cue to reference, and must learn over the second year that the true cue is the caregiver's point of gaze, not their own [6].

Second, while the social model did not outperform the base model on the full corpus (where many words were paired with their referents several times), on a smaller corpus (taking every other utterance), the social cue model did slightly outperform a model without social cues (max F-score=0.43 vs. 0.37). Third, the addition of social cues allowed the model to infer the intent of a speaker even in the absence of a word being used. In the right-hand column of Table 2, we give an example of a situation in which the caregiver simply says "see that?" but from the direction of the infant's eyes and the location of her hand, the model correctly infers that she is talking about the COW, not either of the other possible referents. This kind of inference might lead the way in allowing infants to learn words like pronouns, which serve pick out an unambiguous focus of attention (one that is so obvious based on social and contextual cues that it does not need to be named). Finally, in the next section we show that the addition of social cues to the model allows correct performance in experimental tests of social generalization which only children older than 18 months can pass, suggesting perhaps that the social model is closer to the strategy used by more mature word learners.

Table 2: Intentions inferred by the Bayesian model after having learned a lexicon from the corpus. (IE=Infant's eyes, CH=Caregiver's hands).

| Words | "look at the moocow" | "see the bear by the rattle?" | "see that?" |
|---|---|---|---|
| Objects | COW GIRL BEAR | BEAR RATTLE COW | BEAR RATTLE COW |
| Social Cues | | | IE & CH→COW |
| Inferred intention | COW | BEAR RATTLE | COW |

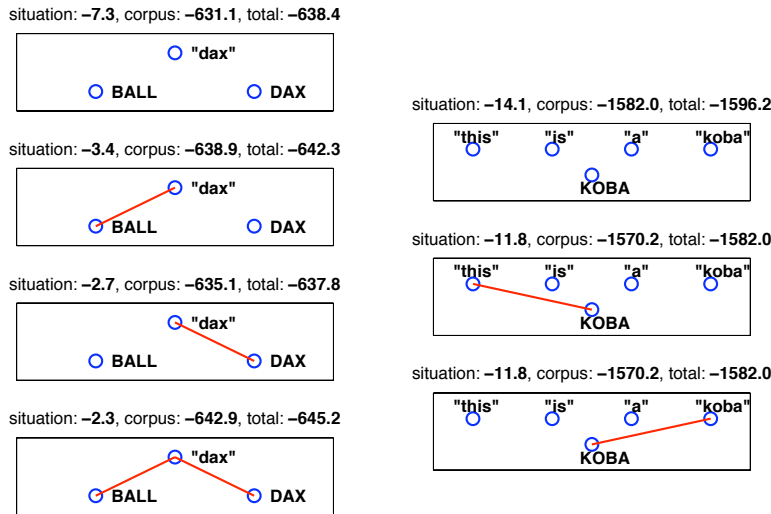

situation: **–7.3**, corpus: **–631.1**, total: **–638.4**

situation: **–3.4**, corpus: **–638.9**, total: **–642.3**

situation: **–2.7**, corpus: **–635.1**, total: **–637.8**

situation: **–2.3**, corpus: **–642.9**, total: **–645.2**

situation: **–14.1**, corpus: **–1582.0**, total: **–1596.2**

situation: **–11.8**, corpus: **–1570.2**, total: **–1582.0**

situation: **–11.8**, corpus: **–1570.2**, total: **–1582.0**

Figure 4: Possible outcomes in (right) a mutual-exclusivity situation and (left) a fast-mapping situation. Situation score is the log probability of the situation (blue dots represent words and objects) under a lexicon (mappings are red lines). Corpus score is the posterior log likelihood of the entire old corpus, including both prior and likelihood terms.

## 5   Coverage of experimental phenomena

**Mutual exclusivity.** When children as young as sixteen months hear a request for a novel word (e.g. where is the dax?) they make a surprising inference: they conclude that the novel word applies to a novel object[7, 8]. This inference is surprising because there seems to be no *prima facie* reason why children should make it—after all, why shouldnt dax simply be another name for a ball? The experimental phenomenon of "mutual exclusivity" has become a touchstone for theories of word-learning: while some authors argue that children use a piece of language-specific knowledge, a principle of mutual exclusivity (that objects do not have two labels), to make this inference [7], others have argued that childrens mapping of the novel noun is a consequence of more general social-pragmatic principles [9].

We test whether, instead of following from language-specific knowledge or pragmatic principles, the same inference could simply be a result of the probabilistic structure of our model. We use the model to infer the best lexicon for a simple artificial corpus (similar to that used in [10]). We then present the model with a single new situation, analogous to the mutual exclusivity experiments (left side of Figure 4). This new situation consists of hearing a novel word ("dax") and seeing both a familiar object and a novel object (BALL and DAX). We then compare four different lexicons on their coverage of both this situation and the previous corpus: (1) one that learns nothing new from the new situation, (2) one that maps dax to BALL, (3) one that maps "dax" to DAX, and (4) one that maps dax to both.

We evaluate the scores of these lexicons on both the new situation and the old corpus. While learning both words produces the best score on the new situation, explaining with high probability why the word "dax" was produced, it performs worst on the rest of the corpus. In particular, it gives a low probability to the coincidence that, while "dax" meant BALL the entire time, the model happened never to hear dax when there was a ball around. In contrast, a lexicon learning no new words scores best on the corpus (because of the prior on smaller lexicons) but has no explanation for why it heard the word "dax" in the new situation. The lexicon which learns "dax"→BALL scores well on neither the corpus nor the new situation: it has no explanation for why it never heard "dax" before, but it also must take into account the fact that "dax" is only half as likely to be spoken when a BALL is present because the word "ball" also could have been produced.

Thus, the correct lexicon, which learns that dax→DAX, performs best when taking into account both the current situation and the model's prior experience. The success of this lexicon (robust across a variety of simulations and parameter settings) suggests that explaining the phenomenon of mutual exclusivity may not require appeals to special principles, either pragmatic or language-specific. Instead, the mutual exclusivity phenomenon may come from a general goal: to learn the lexicon which best explains the utterances the learner hears, given their context.

situation: **−10.2**, corpus: **−772.2**, total: **−782.4**

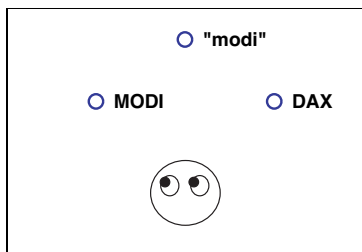

situation: **−6.2**, corpus: **−774.2**, total: **−780.3**

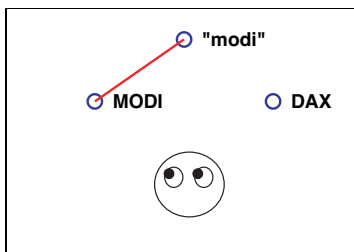

Figure 5: Possible outcomes in a social generalization experiment. The eye-gaze of the speaker (pointing to the MODI) is the only cue which determines that the word "modi" should be mapped to the MODI object; despite this, our model finds the correct mapping.

situation: **−9.1**, corpus: **−774.2**, total: **−783.2**

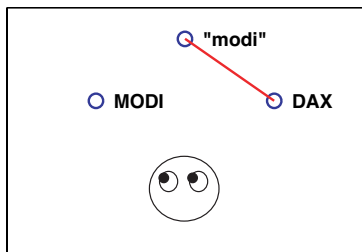

situation: **−6.1**, corpus: **−776.2**, total: **−782.3**

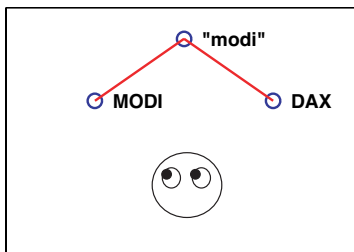

**Fast-mapping.** A second phenomenon which has been much discussed in the psychological literature is fast-mapping [11]. This label refers to the ability of older children to learn a novel label for a novel object in a well-understood sentence frame after only one or a few exposures and retain it over a significant delay. There are two surprising components to this task: first, the ability of children to learn from a single exposure, and second, the retention of the word for a long period. Although our model cannot speak to the retention interval, our non-social model predicts that a single, ambiguous situation can give enough evidence to learn a new word.

Our scenario is similar to the experimental setup used by Markson and Bloom [12]. We learn a lexicon for a small artificial corpus that contains some number of function words, which do not co-occur regularly with any object. We then present the model with a new situation in which there is a novel referent, three words that had been "function words" in our corpus, and one new word (analogous to seeing a novel object, a KOBA, and hearing the utterance "this is a Koba!"; see Figure 4 right side for details). In this scenario, the model strongly favors learning "koba"→KOBA. If it learns nothing, it is penalized on its inability to explain the new situation; if it learns a mapping to a function word, it must explain why this function word was not used referentially in the rest of its experience. Thus, when the other words in the utterance are familiar, our model will learn an appropriate lexical mapping from even a single situation.

**Social generalization.** By adding the ability to learn social cues, our model gains the ability to learn words even in fully ambiguous situations. An experimental demonstration of this phenomena with children is given by Hollich, Hirsh-Pasek, and Golinkoff [6]. In one study, they showed children two novel objects while an experimenter said "Look at the modi!" and looked directly at one of the objects. While 12-month-olds were not able to learn that the word "modi" mapped onto the object that the experimenter looked at, both 18- and 24-month-olds correctly made this inference.

As pictured in Figure 5, our model shows this same pattern of inference. While the best explanation of this situation was given by assuming that the word "modi" mapped to both novel objects (bottom right), this alternative was not preferred because it added two mappings to the lexicon rather than one. On the other hand, the most parsimonious option according to the prior was not to learn any new words, but this did not account for the new evidence. Of the two remaining options, the mapping of "modi" to the correct object was preferred exclusively on account of the distribution of social cues. Much like the older children in Hollich and colleagues' experiment, our model was first able to learn the relevance of particular social cues over the course of its experience (e.g., by processing the corpus) and then apply this knowledge in a novel, ambiguous situation (Figure 5).

# 6 Conclusions

We have presented a Bayesian model of cross-situational world-learning which outperforms both baseline associative models and a more sophisticated translation model on learning from noisy corpus data. However, the strength of our model is not just its performance on the corpus, but also a more natural formulation which may contribute to the clarity of our understanding of word learning.

By organizing our model around determining the speaker's referential intent, we find that several puzzling empirical phenomena in word-learning can be explained as consequences of the structure of the model. The first is mutual exclusivity, the tendency to avoid mapping a novel word to a familiar object when a novel object is available. Researchers in the psychological literature have attempted to explain this type of phenomenon in terms of both language-specific constraints and more general social principles. We suggest, however, that mutual exclusivity may be explained as one of a variety of rational inferences that word-learners can make when presented with an ambiguous situation. The same principle applies to the phenomenon of fast-mapping: given the evidence against other mappings, a rational word-learner would do best to learn the novel mapping. In both of these cases, the relevant phenomena come from the basic model design and domain general principles of inference; as do, for instance, the taxonomic inferences observed by Xu & Tenenbaum [13].

Because it is based on a well-posed generative process, the model can be easily extended to account for joint learning with other domains. We have illustrated this by giving an extension to our basic model of social intention, in which social cues independently contribute to establishing the focus of referential intention in a particular situation. A strength of this extension is that the model does not need to know before learning which cues are relevant for establishing referential intention (and indeed, these cues may vary across cultures where pointing is accomplished in different ways). While Yu and Ballard [4] modify their model to incorporate the focus of intention, their social model assumes that the socially-salient objects are externally indicated—it cannot learn what cues signal that focus or their relative weights. Using these learned social cues our model is able to succeed in learning words even when there is no consistent pattern of co-occurrece (either because of a lack of data or because of a truly ambiguous situation).

This brings us to the question of the psychological status of our model. Our model does not embody a theory about the process or algorithm that children follow to learn words. Instead, our model can been seen as a proposal about the representations and principles underlying word-learning. According to this proposal, it is not necessary to represent association probabilities for all word-concept pairs in order to learn words statistically. Instead, learners can learn a lexicon consisting only of guesses about the meanings of words. And by applying principles of probabilistic inference to this lexicon, it may be possible to bootstrap into the broader social, communicative system.

## Footnotes

[1]Note that, since we ignore word order, the distribution of words in a sentence should be exchangeable given the lexicon and situation. This implies, by de Finetti's theorem, that they are independent conditioned on a latent state—we assume that the latent state giving rise to words is the *intention* of the speaker.

[2] In order to speed convergence we used a simulated tempering scheme with three temperature chains and a range of data-driven proposals.

# References

[1] S. Pinker. *Learnability and cognition: the acquisition of argument structure*. MIT Press, 1989.

[2] L. Gleitman. The structural sources of verb meanings. *Language acquisition*, 1:3–55, 1990.

[3] J.M. Siskind. A computational study of cross-situational techniques for learning word-to-meaning mappings. *Cognition*, 61(1):39–91, 1996.

[4] C. Yu and D. Ballard. A unified model of word learning: Integrating statistical and social cues. *Neurocomputing*, in press.

[5] B. MacWhinney. *The CHILDES Project: Tools for Analyzing Talk*. Lawrence Erlbaum, 2000.

[6] G. Hollich, K. Hirsh-Pasek, and R.M. Golinkoff. II. The Emergentist Coalition Model. *Monographs of the Society for Research in Child Development*, 65(3):17–29, 2000.

[7] E.M. Markman. *Categorization and Naming in Childern: problems of induction*. Bradford Book, 1989.

[8] C.B. Mervis and J. Bertrand. Acquisition of the Novel Name-Nameless Category (N3C) Principle. *Child Development*, 65(6):1646–1662, 1994.

[9] E. V. Clark. On the logic of contrast. *Journal of Child Language*, 15:317–335, 1988.

[10] C Yu and L Smith. Rapid word learning under uncertainty via cross-situational statistics. *Psychological Science*, in press.

[11] S. Carey. The child as word learner. In *Linguistic theory and psychological reality*. MA: MIT Press, 1978.

[12] L. Markson and P. Bloom. Evidence against a dedicated system for word learning in children. *Nature*, 385(6619):813–815, 1997.

[13] F. Xu and J. B. Tenenbaum. Word learning as bayesian inference. *Psychological Review*, 2007.

